# Neural Representation of Multi-Dimensional Stimuli

**Christian W. Eurich, Stefan D. Wilke and Helmut Schwegler**
Institut für Theoretische Physik
Universität Bremen, Germany
*(eurich,swilke,schwegler)@physik.uni-bremen.de*

## Abstract

The encoding accuracy of a population of stochastically spiking neurons is studied for different distributions of their tuning widths. The situation of identical radially symmetric receptive fields for all neurons, which is usually considered in the literature, turns out to be disadvantageous from an information-theoretic point of view. Both a variability of tuning widths and a fragmentation of the neural population into specialized subpopulations improve the encoding accuracy.

## 1 Introduction

The topic of neuronal tuning properties and their functional significance has focused much attention in the last decades. However, neither empirical findings nor theoretical considerations have yielded a unified picture of optimal neural encoding strategies given a sensory or motor task. More specifically, the question as to whether narrow tuning or broad tuning is advantageous for the representation of a set of stimulus features is still being discussed. Empirically, both situations are encountered: small receptive fields whose diameter is less than one degree can, for example, be found in the human retina [7], and large receptive fields up to 180° in diameter occur in the visual system of tongue-projecting salamanders [10]. On the theoretical side, arguments have been put forward for small [8] as well as for large [5, 1, 9, 3, 13] receptive fields.

In the last years, several approaches have been made to calculate the encoding accuracy of a neural population as a function of receptive field size [5, 1, 9, 3, 13]. It has turned out that for a firing rate coding, large receptive fields are advantageous provided that $D \geq 3$ stimulus features are encoded [9, 13]. For binary neurons, large receptive fields are advantageous also for $D = 2$ [5, 3].

However, so far only radially symmetric tuning curves have been considered. For neural populations which lack this symmetry, the situation may be very different. Here we study the encoding accuracy of a population of stochastically spiking neurons. A Fisher information analysis performed on different distributions of tunings widths will indeed reveal a much more detailed picture of neural encoding strategies.

## 2 Model

Consider a $D$-dimensional stimulus space, $X$. A stimulus is characterized by a position $\mathbf{x} = (x_1, \ldots, x_D) \in X$, where the value of feature $i$, $x_i$ ($i = 1, \ldots, D$), is measured relative to the total range of values in the $i$-th dimension such that it is dimensionless. Information about the stimulus is encoded by a population of $N$ stochastically spiking neurons. They are assumed to have independent spike generation mechanisms such that the joint probability distribution for observing $\mathbf{n} = (n^{(1)}, \ldots, n^{(k)}, \ldots, n^{(N)})$ spikes within a time interval $\tau$, $P_s(\mathbf{n}; \mathbf{x})$, can be written in the form

$$P_s(\mathbf{n}; \mathbf{x}) = \prod_{k=1}^{N} P_s^{(k)}(n^{(k)}; \mathbf{x}), \tag{1}$$

where $P_s^{(k)}(n^{(k)}; \mathbf{x})$ is the single-neuron probability distribution of the number of observed spikes given the stimulus at position $\mathbf{x}$. Note that (1) does not exclude a correlation of the neural firing rates, i.e., the neurons may have common input or even share the same tuning function.

The firing rates depend on the stimulus via the local values of the tuning functions, such that $P_s^{(k)}(n^{(k)}; \mathbf{x})$ can be written in the form $P_s^{(k)}(n^{(k)}; \mathbf{x}) = S\left(n^{(k)}, f^{(k)}(\mathbf{x}), \tau\right)$, where the tuning function of neuron $k$, $f^{(k)}(\mathbf{x})$, gives its mean firing rate in response to the stimulus at position $\mathbf{x}$. We assume here a form of the tuning function that is not necessarily radially symmetric,

$$f^{(k)}(\mathbf{x}) = F\phi\left(\sum_{i=1}^{D} \frac{(x_i - c_i^{(k)})^2}{\sigma_i^{(k)2}}\right) =: F\phi\left(\xi^{(k)2}\right), \tag{2}$$

where $\mathbf{c}^{(k)} = (c_1^{(k)}, \ldots, c_D^{(k)})$ is the center of the tuning curve of neuron $k$, $\sigma_i^{(k)}$ is its tuning width in the $i$-th dimension, $\xi_i^{(k)2} := (x_i - c_i^{(k)})^2/\sigma_i^{(k)2}$ for $i = 1, \ldots, D$, and $\xi^{(k)2} := \xi_1^{(k)2} + \ldots + \xi_D^{(k)2}$. $F > 0$ denotes the maximal firing rate of the neurons, which requires that $\max_{z \geq 0} \phi(z) = 1$.

We assume that the tuning widths $\sigma_1^{(k)}, \ldots, \sigma_D^{(k)}$ of each neuron $k$ are drawn from a distribution $P_\sigma(\sigma_1, \ldots, \sigma_D)$. For a population of tuning functions with centers $\mathbf{c}^{(1)}, \ldots, \mathbf{c}^{(N)}$, a density $\eta(\mathbf{x})$ is introduced according to $\eta(\mathbf{x}) := \sum_{k=1}^{N} \delta(\mathbf{x} - \mathbf{c}^{(k)})$.

The encoding accuracy can be quantified by the Fisher information matrix, $\mathbf{J}$, which is defined as

$$J_{ij}(\mathbf{x}) := E\left[\left(\frac{\partial}{\partial x_i} \ln P(\mathbf{n}; \mathbf{x})\right)\left(\frac{\partial}{\partial x_j} \ln P(\mathbf{n}; \mathbf{x})\right)\right], \tag{3}$$

where $E[\ldots]$ denotes the expectation value over the probability distribution $P(\mathbf{n}; \mathbf{x})$ [2]. The Fisher information yields a lower bound on the expected error of an unbiased estimator that retrieves the stimulus $\mathbf{x}$ from the noisy neural activity (Cramér-Rao inequality) [2]. The minimal estimation error for the $i$-th feature $x_i$, $\epsilon_{i,\min}$, is given by $\epsilon_{i,\min}^2 = (\mathbf{J}^{-1})_{ii}$ which reduces to $\epsilon_{i,\min}^2 = 1/J_{ii}(\mathbf{x})$ if $\mathbf{J}$ is diagonal.

We shall now derive a general expression for the population Fisher information. In the next chapter, several cases and their consequences for neural encoding strategies will be discussed.

For model neuron $(k)$, the Fisher information (3) reduces to

$$J_{ij}^{(k)}(\mathbf{x}; \sigma_1^{(k)}, \ldots, \sigma_D^{(k)}) = \frac{1}{\sigma_i^{(k)} \sigma_j^{(k)}} A_\phi\left(\xi^{(k)2}, F, \tau\right) \xi_i^{(k)} \xi_j^{(k)}, \tag{4}$$

where the dependence on the tuning widths is indicated by the list of arguments. The function $A_\phi$ depends on the shape of the tuning function and is given in [13]. The independence assumption (1) implies that the population Fisher information is the sum of the contributions of the individual neurons, $\sum_{k=1}^{N} J_{ij}^{(k)}(\mathbf{x}; \sigma_1^{(k)}, \ldots, \sigma_D^{(k)})$. We now define a population Fisher information which is averaged over the distribution of tuning widths $P_\sigma(\sigma_1, \ldots, \sigma_D)$:

$$\langle J_{ij}(\mathbf{x})\rangle_\sigma = \sum_{k=1}^{N} \int d\sigma_1 \ldots d\sigma_D \, P_\sigma(\sigma_1, \ldots, \sigma_D) \, J_{ij}^{(k)}(\mathbf{x}; \sigma_1, \ldots, \sigma_D). \qquad (5)$$

Introducing the density of tuning curves, $\eta(\mathbf{x})$, into (5) and assuming a constant distribution, $\eta(\mathbf{x}) \equiv \eta \equiv$ const., one obtains the result that the population Fisher information becomes independent of $\mathbf{x}$ and that the off-diagonal elements of $\mathbf{J}$ vanish [13]. The average population Fisher information then becomes

$$\langle J_{ij}\rangle_\sigma = \eta D K_\phi(F, \tau, D) \left\langle \frac{\prod_{l=1}^{D} \sigma_l}{\sigma_i^2} \right\rangle_\sigma \delta_{ij}, \qquad (6)$$

where $K_\phi$ depends on the geometry of the tuning curves and is defined in [13].

## 3  Results

In this section, we consider different distributions of tuning widths in (6) and discuss advantageous and disadvantageous strategies for obtaining a high representational accuracy in the neural population.

**Radially symmetric tuning curves.**  For radially symmetric tuning curves of width $\overline{\sigma}$, the tuning-width distribution reads

$$P_\sigma(\sigma_1, \ldots, \sigma_D) = \prod_{i=1}^{D} \delta(\sigma_i - \overline{\sigma});$$

see Fig. 1a for a schematic visualization of the arrangement of the tuning widths for the case $D = 2$. The average population Fisher information (6) for $i = j$ becomes

$$\langle J_{ii}\rangle_\sigma = \eta D K_\phi(F, \tau, D) \overline{\sigma}^{D-2}, \qquad (7)$$

a result already obtained by Zhang and Sejnowski [13]. Equation (7) basically shows that the minimal estimation error increases with $\overline{\sigma}$ for $D = 1$, that it does not depend on $\overline{\sigma}$ for $D = 2$, and that it decreases as $\overline{\sigma}$ increases for $D \geq 3$. We shall discuss the relevance of this case below.

**Identical tuning curves without radial symmetry.**  Next we discuss tuning curves which are identical but not radially symmetric; the tuning-width distribution for this case is

$$P_\sigma(\sigma_1, \ldots, \sigma_D) = \prod_{i=1}^{D} \delta(\sigma_i - \overline{\sigma}_i),$$

where $\overline{\sigma}_i$ denotes the fixed width in dimension $i$. For $i = j$, the average population Fisher information (6) reduces to [11, 4]

$$\langle J_{ii}\rangle_\sigma = \eta D K_\phi(F, \tau, D) \frac{\prod_{l=1}^{D} \overline{\sigma}_l}{\overline{\sigma}_i^2}. \qquad (8)$$

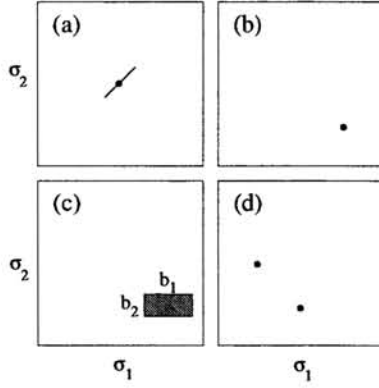

Figure 1: Visualization of different distributions of tuning widths for $D = 2$. (a) Radially symmetric tuning curves. The dot indicates a fixed $\bar{\sigma}$, while the diagonal line symbolizes a variation in $\bar{\sigma}$ discussed in [13]. (b) Identical tuning curves which are not radially symmetric. (c) Tuning widths uniformly distributed within a small rectangle. (d) Two subpopulations each of which is narrowly tuned in one dimension and broadly tuned in the other direction.

Equation (8) contains (7) as a special case. From (8) it becomes immediately clear that the expected minimal square encoding error for the $i$-th stimulus feature, $\epsilon^2_{i,\min} = 1/\langle J_{ii}(\mathbf{x})\rangle_\sigma$, depends on $i$, i. e., *the population specializes in certain features*. The error obtained in dimension $i$ thereby depends on the tuning widths in all dimensions.

Which encoding strategy is optimal for a population whose task it is to encode a single feature, say feature $i$, with high accuracy while not caring about the other dimensions? In order to answer this question, we re-write (8) in terms of receptive field overlap.

For the tuning functions $f^{(k)}(\mathbf{x})$ encountered empirically, large values of the single-neuron Fisher information (4) are typically restricted to a region around the center of the tuning function, $\mathbf{c}^{(k)}$. The fraction $p(\beta)$ of the Fisher information that falls into a region $E_D :$ $\sqrt{\xi^{(k)2}} \leq \beta$ around $\mathbf{c}^{(k)}$ is given by

$$p(\beta) := \frac{\int\limits_{E_D} \mathrm{d}^D x \sum_{i=1}^{D} J_{ii}^{(k)}(\mathbf{x})}{\int\limits_{X} \mathrm{d}^D x \sum_{i=1}^{D} J_{ii}^{(k)}(\mathbf{x})} = \frac{\int\limits_{0}^{\beta} \mathrm{d}\xi\, \xi^{D+1} A_\phi(\xi^2, F, \tau)}{\int\limits_{0}^{\infty} \mathrm{d}\xi\, \xi^{D+1} A_\phi(\xi^2, F, \tau)}, \tag{9}$$

where the index $(k)$ was dropped because the tuning curves are assumed to have identical shapes. Equation (9) allows the definition of an effective receptive field, $\mathrm{RF}_{\mathrm{eff}}^{(k)}$, inside of which neuron $k$ conveys a major fraction $p_0$ of Fisher information, $\mathrm{RF}_{\mathrm{eff}}^{(k)} := \left\{ \mathbf{x} \middle| \sqrt{\xi^{(k)2}} \leq \beta_0 \right\}$, where $\beta_0$ is chosen such that $p(\beta_0) = p_0$. The Fisher information a neuron $k$ carries is small unless $\mathbf{x} \in \mathrm{RF}_{\mathrm{eff}}^{(k)}$. This has the consequence that a fixed stimulus $\mathbf{x}$ is actually encoded only by a subpopulation of neurons. The point $\mathbf{x}$ in stimulus space is covered by

$$N_{\mathrm{code}} := \eta \frac{2\pi^{D/2}(\beta_0)^D}{D\Gamma(D/2)} \prod_{j=1}^{D} \bar{\sigma}_j \tag{10}$$

receptive fields. With the help of (10), the average population Fisher information (8) can be re-written as

$$\langle J_{ii}\rangle_\sigma = \frac{D^2\Gamma(D/2)}{2\pi^{D/2}(\beta_0)^D} K_\phi(F, \tau, D) \frac{N_{\mathrm{code}}}{\bar{\sigma}_i^2}. \tag{11}$$

Equation (11) can be interpreted as follows: We assume that the population of neurons encodes stimulus dimension $i$ accurately, while all other dimensions are of secondary importance. The average population Fisher information for dimension $i$, $\langle J_{ii}\rangle_\sigma$, is determined by the tuning width in dimension $i$, $\bar{\sigma}_i$, and by the size of the active subpopulation, $N_{\mathrm{code}}$. There is a tradeoff between these quantities. On the one hand, the encoding error can be decreased by decreasing $\bar{\sigma}_i$, which enhances the Fisher information carried by each single

neuron. Decreasing $\overline{\sigma}_i$, on the other hand, will also shrink the active subpopulation via (10). This impairs the encoding accuracy, because the stimulus position is evaluated from the activity of fewer neurons. If (11) is valid due to a sufficient receptive field overlap, $N_{\text{code}}$ can be increased by increasing the tuning widths, $\overline{\sigma}_j$, in all other dimensions $j \neq i$. This effect is illustrated in Fig. 2 for $D = 2$.

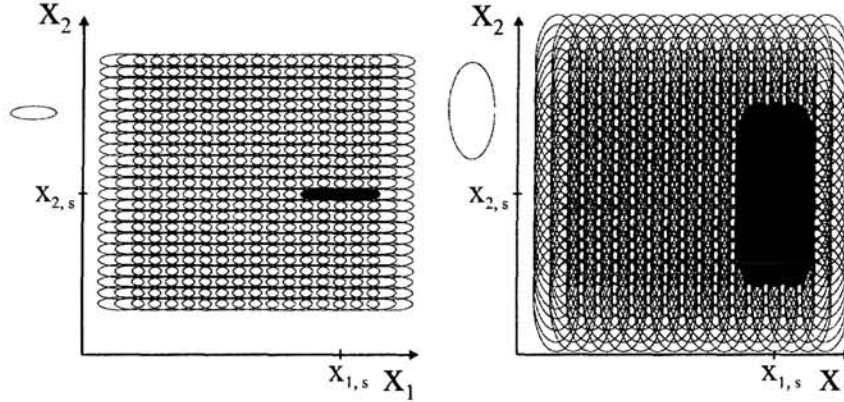

Figure 2: Encoding strategy for a stimulus characterized by parameters $x_{1,\text{s}}$ and $x_{2,\text{s}}$. Feature $x_1$ is to be encoded accurately. Effective receptive field shapes are indicated for both populations. If neurons are narrowly tuned in $x_2$ (left), the active population (solid) is small (here: $N_{\text{code}} = 3$). Broadly tuned receptive fields for $x_2$ (right) yield a much larger population (here: $N_{\text{code}} = 27$) thus increasing the encoding accuracy.

It shall be noted that although a narrow tuning width $\overline{\sigma}_i$ is advantageous, the limit $\overline{\sigma}_i \longrightarrow 0$ yields a bad representation. For narrowly tuned cells, gaps appear between the receptive fields: The condition $\eta(\mathbf{x}) \equiv$ const. breaks down, and (6) is no longer valid. A more detailed calculation shows that the encoding error diverges as $\overline{\sigma}_i \longrightarrow 0$ [4]. The fact that the encoding error decreases for both narrow tuning and broad tuning – due to (11) – proves the existence of an *optimal tuning width*. An example is given in Fig. 3a.

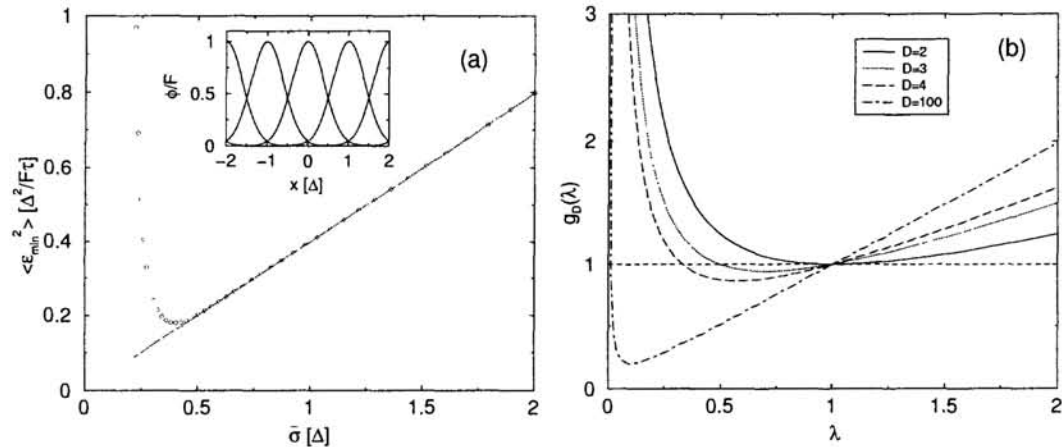

Figure 3: (a) Example for the encoding behavior with narrow tuning curves arranged on a regular lattice of dimension $D = 1$ (grid spacing $\Delta$). Tuning curves are Gaussian, and neural firing is modeled as a Poisson process. Dots indicate the minimal square encoding error averaged over a uniform distribution of stimuli, $\langle \epsilon_{\text{min}}^2 \rangle$, as a function of $\overline{\sigma}$. The minimum is clearly visible. The dotted line shows the corresponding approximation according to (8). The inset shows Gaussian tuning curves of optimal width, $\overline{\sigma}^{\text{opt}} \approx 0.4\Delta$. (b) $g_D(\lambda)$ as a function of $\lambda$ for different values of $D$.

**Narrow distribution of tuning curves.** In order to study the effects of encoding the stimulus with distributed tuning widths instead of identical tuning widths as in the previous cases, we now consider the distribution

$$P_\sigma(\sigma_1,\ldots,\sigma_D) = \prod_{i=1}^{D} \frac{1}{b_i}\Theta\left[\sigma_i - \left(\overline{\sigma}_i - \frac{b_i}{2}\right)\right]\Theta\left[\left(\overline{\sigma}_i + \frac{b_i}{2}\right) - \sigma_i\right], \qquad (12)$$

where $\Theta$ denotes the Heaviside step function. Equation (12) describes a uniform distribution in a $D$-dimensional cuboid of size $b_1,\ldots,b_D$ around $(\overline{\sigma}_1,\ldots\overline{\sigma}_D)$; cf. Fig. 1c. A straightforward calculation shows that in this case, the average population Fisher information (6) for $i = j$ becomes

$$\langle J_{ii}\rangle_\sigma = \eta D K_\phi(F,\tau,D)\frac{\prod_{l=1}^{D}\overline{\sigma}_l}{\overline{\sigma}_i^2}\left\{1 + \frac{1}{12}\left(\frac{b_i}{\overline{\sigma}_i}\right)^2 + \mathcal{O}\left[\left(\frac{b_i}{\overline{\sigma}_i}\right)^4\right]\right\}. \qquad (13)$$

A comparison with (8) yields the astonishing result that an increase in $b_i$ results in an increase in the $i$-th diagonal element of the average population Fisher information matrix and thus in an improvement in the encoding of the $i$-th stimulus feature, while the encoding in dimensions $j \neq i$ is not affected. Correspondingly, the total encoding error can be decreased by increasing an arbitrary number of edge lengths of the cube. *The encoding by a population with a variability in the tuning curve geometries as described is more precise than that by a uniform population. This is true for arbitrary $D$.* Zhang and Sejnowski [13] consider the more artificial situation of a correlated variability of the tuning widths: tuning curves are always assumed to be radially symmetric. This is indicated by the diagonal line in Fig. 1a. A distribution of tuning widths restricted to this subset yields an average population Fisher information $\propto \langle \overline{\sigma}^{D-2}\rangle$ and does not improve the encoding for $D = 2$ or $D = 3$.

**Fragmentation into $D$ subpopulations.** Finally, we study a family of distributions of tuning widths which also yields a lower minimal encoding error than the uniform population. Let the density of tuning curves be given by

$$P_\sigma(\sigma_1,\ldots,\sigma_D) = \frac{1}{D}\sum_{i=1}^{D}\delta(\sigma_i - \lambda\overline{\sigma})\prod_{j\neq i}\delta(\sigma_j - \overline{\sigma}), \qquad (14)$$

where $\lambda > 0$. For $\lambda = 1$, the population is uniform as in (7). For $\lambda \neq 1$, the population is split up into $D$ subpopulations; in subpopulation $i$, $\sigma_i$ is modified while $\sigma_j \equiv \overline{\sigma}$ for $j \neq i$. See Fig. 1d for an example. The diagonal elements of the average population Fisher information are

$$\langle J_{ii}\rangle_\sigma = \eta D K_\phi(F,\tau,D)\overline{\sigma}^{D-2}\left\{\frac{1+(D-1)\lambda^2}{D\lambda}\right\}, \qquad (15)$$

where the term in brackets will be abbreviated as $g_D(\lambda)$. $\langle J_{ii}\rangle_\sigma$ does not depend on $i$ in this case because of the symmetry in the subpopulations. Equation (15) and the uniform case (7) differ by $g_D(\lambda)$ which will now be discussed. Figure 3b shows $g_D(\lambda)$ for different values of $D$. For $\lambda = 1$, $g_D(\lambda) = 1$ and (7) is recovered as expected. $g_D(\lambda) = 1$ also holds for $\lambda = 1/(D-1) < 1$: narrowing one tuning width in each subpopulation will at first decrease the resolution provided $D \geq 3$; this is due to the fact that $N_{\text{code}}$ is decreased. For $\lambda < 1/(D-1)$, however, $g_D(\lambda) > 1$, and the resolution exceeds $\langle J_{ii}\rangle_\sigma$ in (7) because each neuron in the $i$-th subpopulation carries a high Fisher information in the $i$-th dimension. $D = 2$ is a special case where no impairment of encoding occurs because the effect of a decrease of $N_{\text{code}}$ is less pronounced. Interestingly, an increase in $\lambda$ also yields an improvement in the encoding accuracy. This is a combined effect resulting from an increase in $N_{\text{code}}$ on the one hand and the existence of $D$ subpopulations, $D - 1$ of

which maintain their tuning widths in each dimension on the other hand. The discussion of $g_D(\lambda)$ leads to the following encoding strategy. For small $\lambda$, $\langle J_{ii} \rangle_\sigma$ increases rapidly, which suggests a fragmentation of the population into $D$ subpopulations each of which encodes one feature with high accuracy, i.e., one tuning width in each subpopulation is small whereas the remaining tuning widths are broad. Like in the case discussed above, the theoretical limit of this method is a breakdown of the approximation of $\eta \equiv$ const. and the validity of (6) due to insufficient receptive field overlap.

## 4 Discussion and Outlook

We have discussed the effects of a variation of the tuning widths on the encoding accuracy obtained by a population of stochastically spiking neurons. The question of an optimal tuning strategy has turned out to be more complicated than previously assumed. More specifically, the case which focused most attention in the literature – radially symmetric receptive fields [5, 1, 9, 3, 13] – yields a worse encoding accuracy than most other cases we have studied: uniform populations with tuning curves which are not radially symmetric; distributions of tuning curves around some symmetric or non-symmetric tuning curve; and the fragmentation of the population into $D$ subpopulations each of which is specialized in one stimulus feature.

In a next step, the theoretical results will be compared to empirical data on encoding properties of neural populations. One aspect is the existence of sensory maps which consist of neural subpopulations with characteristic tuning properties for the features which are represented. For example, receptive fields of auditory neurons in the midbrain of the barn owl have elongated shapes [6]. A second aspect concerns the short-term dynamics of receptive fields. Using single-unit recordings in anaesthetized cats, Wörgötter et al. [12] observed changes in receptive field size taking place in 50–100 ms. Our findings suggest that these dynamics alter the resolution obtained for the corresponding stimulus features. The observed effect may therefore realize a mechanism of an adaptable selective signal processing.

## References

[1] Baldi, P. & Heiligenberg, W. (1988) *Biol. Cybern.* **59**:313–318.

[2] Deco, G. & Obradovic, D. (1997) *An Information-Theoretic Approach to Neural Computing.* New York: Springer.

[3] Eurich, C. W. & Schwegler, H. (1997) *Biol. Cybern.* **76**: 357–363.

[4] Eurich, C. W. & Wilke, S. D. (2000) *Neural Comp.* (in press).

[5] Hinton, G. E., McClelland, J. L. & Rumelhart, D. E (1986) In Rumelhart, D. E. & McClelland, J. L. (eds.), *Parallel Distributed Processing, Vol. 1*, pp. 77–109. Cambridge MA: MIT Press.

[6] Knudsen, E. I. & Konishi, M. (1978) *Science* **200**:795–797.

[7] Kuffler, S. W. (1953) *J. Neurophysiol.* **16**:37–68.

[8] Lettvin, J. Y., Maturana, H. R., McCulloch, W. S. & Pitts, W. H. (1959) *Proc. Inst. Radio Eng. NY* **47**:1940–1951.

[9] Snippe, H. P. & Koenderink, J. J. (1992) *Biol. Cybern.* **66**:543–551.

[10] Wiggers, W., Roth, G., Eurich, C. W. & Straub, A. (1995) *J. Comp. Physiol. A* **176**:365–377.

[11] Wilke, S. D. & Eurich, C. W. (1999) In Verleysen, M. (ed.), *ESANN 99, European Symposium on Artificial Neural Networks*, pp. 435–440. Brussels: D-Facto.

[12] Wörgötter, F., Suder, K., Zhao, Y., Kerscher, N., Eysel, U. T. & Funke, K. (1998) *Nature* **396**:165–168.

[13] Zhang, K. & Sejnowski, T. J. (1999) *Neural Comp.* **11**:75–84.
